# Learning to categorize objects using temporal coherence

Suzanna Becker*
The Rotman Research Institute
Baycrest Center
3560 Bathurst St.
Toronto, Ontario, M6A 2E1

## Abstract

The invariance of an objects' identity as it transformed over time provides a powerful cue for perceptual learning. We present an unsupervised learning procedure which maximizes the mutual information between the representations adopted by a feed-forward network at consecutive time steps. We demonstrate that the network can learn, entirely unsupervised, to classify an ensemble of several patterns by observing pattern trajectories, even though there are abrupt transitions from one object to another between trajectories. The same learning procedure should be widely applicable to a variety of perceptual learning tasks.

## 1    INTRODUCTION

A promising approach to understanding human perception is to try to model its developmental stages. There is ample evidence that much of perception is learned. Even some very low level perceptual abilities such as stereopsis (Held, Birch and Gwiazda, 1980; Birch, Gwiazda and Held, 1982) are not present at birth, and appear to be learned. Once rudimentary feature detection abilities have been established, the infant can learn to segment the sensory input, and eventually classify it into familiar patterns. These earliest stages of learning seem to be inherently unsuper-

vised (or "self-supervised"). Gradually, the infant learns to detect regularities in the world. One kind of structure that is ubiquitous in sensory information is spatio-temporal coherence. For example, in speech signals, speaker characteristics such as the fundamental frequency are relatively constant over time. At shorter time scales, individual words are typically composed of long intervals having relatively constant spectral characteristics, corresponding to vowels, with short intervening bursts and rapid transitions corresponding to consonants. The consonants also change across time in very regular ways. This temporal coherence at various scales makes speech predictable, to a certain degree. As one moves about in the world, the visual field flows by in characteristic patterns of expansion, dilation and translation. Since most objects in the visual world move slowly, if at all, the visual scene changes slowly over time, exhibiting the same temporal coherence as other sensory sources. Independently moving rigid objects are invariant with respect to shape, texture and many other features, up to very high level properties such as the object's identity. Even under nonlinear shape distortions, images like clouds drifting across the sky are perceived to have coherent features, in spite of undergoing highly non-rigid transformations. Thus, temporal coherence of the sensory input may provide important cues for segmenting signals in space and time, and for object localization and identification.

## 2 PREVIOUS WORK

A common approach to training neural networks to perform transformation-invariant object recognition is to build in hard constraints which enforce invariance with respect to the transformations of interest. For example, equality constraints among feature-detecting kernels have been used to enforce translation-invariance (Fukushima, 1988; Le Cun et al., 1990). Various other higher-order constraints have been used to enforce viewpoint-invariance (Hinton and Lang, 1985; Zemel, Hinton and Mozer, 1990) and invariance with respect to arbitrary group transformations (Giles and Maxwell, 1987). While in the case of translation-invariance it is straightforward to hard-wire the appropriate constraints, more general linear transformation-invariance requires rather cumbersome machinery, and for arbitrary non-linear transformations the approach is difficult if not impossible.

In contrast to the above approaches, Földiák's model of complex cell development results in translation-invariant orientation detectors without the imposition of any hard constraints (Földiák, 1991). Further, his method is unsupervised. He proposed a modified Hebbian learning rule, in which each weight change depends on the unit's output history:

$$\Delta w_{ij}(t) = \alpha \, \overline{y}_i(t) \, (x_j(t) - w_{ij}(t))$$

where $x_j(t)$ is the activity of the $j$th presynaptic unit at the $t$th time step, and $\overline{y}_i(t)$ is a temporally low-pass filtered trace of the postsynaptic activity of the $i$th unit. Whereas a standard Hebb-rule encourages a unit to detect correlations between its inputs, this rule encourages a unit to produce outputs which are correlated over time. A single unit can therefore learn to group patterns which have zero overlap. Földiák demonstrated this by presenting trajectories of moving lines, with line orientation held constant within each trajectory, to a network whose input features were local orientation detectors. Units became tuned to particular orientations,

independent of location.

While Földiák's work is of interest as a model of cell development in early visual cortex, there are several reasons why it cannot be applied directly to the more general problem of transformation-invariant object recognition. One reason that Földiák's learning rule worked well on the line trajectory problem is that the input representation (oriented line features) made the problem linearly separable: there was no overlap between input features present in successive trajectories, hence it was easy to categorize lines of the same orientation. Generally, in more difficult pattern classification problems (such as digit or speech recognition) the optimal input features cannot be preselected but must be learned, and there is considerable overlap between the component features of different pattern classes. Hence, a multi-layer network is required, and it must be able to optimally select features so as to improve its classification performance. The question of interest here is whether it is possible to train such a network entirely unsupervised? As mentioned above, the temporal coherence of the sensory input may be an important cue for solving this problem in biological systems.

# 3   TEMPORAL-COHERENCE BASED LEARNING

One way to capture the constraint of temporal coherence in a learning procedure is to build it into the objective function. For example, we could try to build representations that are relatively predictable, at least over short time scales. We also need a constraint which captures the notion of high information content; for example, we could require that the network be *unpredictable* over long time scales. A measure which satisfies both criteria is the mutual information between the classifications produced by the network at successive time steps. If the network produces classification $C(t)$ at time $t$ and classification $C(t+1)$ at time $t+1$, the mutual information between the two successive classifications, averaged over the entire sequence of patterns, is given by

$$
\begin{aligned}
I_{C_t;C_{t+1}} &= H(C_t) + H(C_{t+1}) - H(C_t, C_{t+1}) \\
&= -\sum_i \langle p_i{}^t \rangle_t \log \langle p_i{}^t \rangle_t - \sum_j \langle p_j{}^{t+1} \rangle_t \log \langle p_j{}^{t+1} \rangle_t \\
&\quad + \sum_{ij} \langle p_i{}^t p_j{}^{t+1} \rangle_t \log \langle p_i{}^t p_j{}^{t+1} \rangle_t
\end{aligned}
$$

where the angle brackets denote time-averaged quantities.

A set of $n$ output units can be forced to represent a probability distribution over $n$ classes, $C \in \{c_1 \cdots c_n\}$, by adopting states whose probabilities sum to one. This can be done, for example, by using the "softmax" activation function suggested by Bridle (1990):

$$
p_i{}^t = \frac{e^{x_i(t)}}{\sum_{j=1}^{n} e^{x_j(t)}} = P(C(t) = c_i)
$$

where $x_i$ is the total weighted summed input to the $i$th unit, and $p_i{}^t$, the output of the $i$th unit, stands for the probability of the $i$th class, $P(C(t) = c_i)$.

Once we know the probability of the network assigning each pattern to each class, we can compute the mutual information between the classifications produced by the network at neighboring time steps, $C(t)$ and $C(t + 1)$. This requires sampling, over the entire training set, the average probability of each class, as well as the joint probabilities of each possible pair of classifications being produced as successive time steps. The learning involves adjusting the weights in the network so as to maximize the mutual information between the representations produced by the network at adjacent time steps. In the experiments reported here, a gradient ascent procedure was used with the method of conjugate gradients.

One problem with maximizing the information measure described above is that for a fixed amount of entropy in the classifications, $H(C_t)$, the network can always improve the mutual information by decreasing the joint entropy, $H(C_t, C_{t_1})$. In order to achieve low joint entropy, the network must try to assign class probabilities with high certainty, i.e., produce output values near zero or one. Thus the network can always improve its current solution by simply make the weights very large. Unfortunately, this often occurs during learning. To discourage the network from getting stuck in such locally optimal (but very poor) solutions, we introduce a constant $\lambda$ to weight the importance of the joint entropy term in the objective function, so as to maximize the following:

$$I_{C_t;C_{t+1}} \quad = \quad H(C_t) + H(C_{t+1}) - \lambda H(C_t, C_{t+1})$$

In the simulations reported here, we used a value of 0.5 for $\lambda$. This effectively prevents the network from concentrating all its effort on reducing the joint entropy, and forces it to learn more gradually, resulting in more globally optimal solutions.

We have tested this learning procedure on a simple signal classification problem. The pattern set consisted of trajectories of random intensity patterns, drawn from six classes, shown in figure 1. Members of the same class consisted of translated versions of the same pattern, shifted one to five pixels with wrap-around. A trajectory consisted of a block of ten randomly selected patterns from the same class. Between trajectories, the pattern class changed randomly. The network had six input units, twenty hidden units, and six output units. The hidden units used the logistic non-linearity, $\frac{1}{1+e^{-x}}$, and the output units used the softmax activation function. The hidden units had biases but the outputs did not.[1] After training the network on 1200 patterns (20 trajectories of 10 examples of each of the six patterns) for 300 conjugate gradient iterations, the output units always became reasonably specific to particular pattern classes, as shown for a typical run in Figure 2a). The general pattern is that each output unit responds maximally to one or two pattern classes, although some of the units have mixed responses.

This classification problem is extremely difficult for an unsupervised learning procedure, as there is considerable overlap between patterns in different classes, and essentially no overlap between patterns in the same class. It is therefore easy to see why a single unit might end up capturing a few patterns from one class and a few from another. We can create an easier subproblem by only training the network on half the patterns in each class. In this case, the network always learns to separate the six pattern classes either perfectly, or nearly so, as shown in figure 2b).

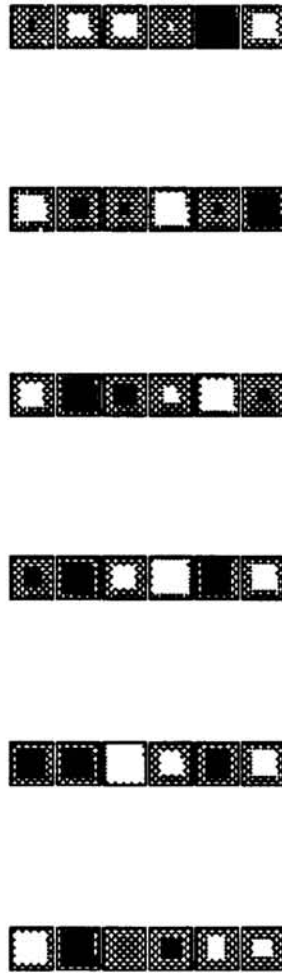

Figure 1: *The set of 6 random patterns used to create pattern trajectories. Each pattern was created by randomly setting the intensities of the 6 pixels, and normalizing the intensity profile to have zero mean.*

## 4  DISCUSSION

Becker and Hinton (1992) showed that a network could learn to extract a continuous parameter of visual scenes which is coherent across space, by maximizing the mutual information between the outputs of two network modules that receive input from spatially adjacent parts of the input. Here, we have shown how the same idea can be applied to the temporal domain, to perform a discrete classification of the input assuming temporal coherence. We could also apply the same algorithm to the problem of unsupervised multi-sensory integration, by forming classifications which are coherent across different sensory modalities, as well as across time.

One advantage of the approach presented here over unsupervised learning procedures such as competitive learning is that units must co-operate to try to find a globally optimal solution. There is therefore incentive for each unit to try to improve the temporal predictability of *all* of the output units' classifications over time, including its own; this discourages any one unit from trying to model all of

a)

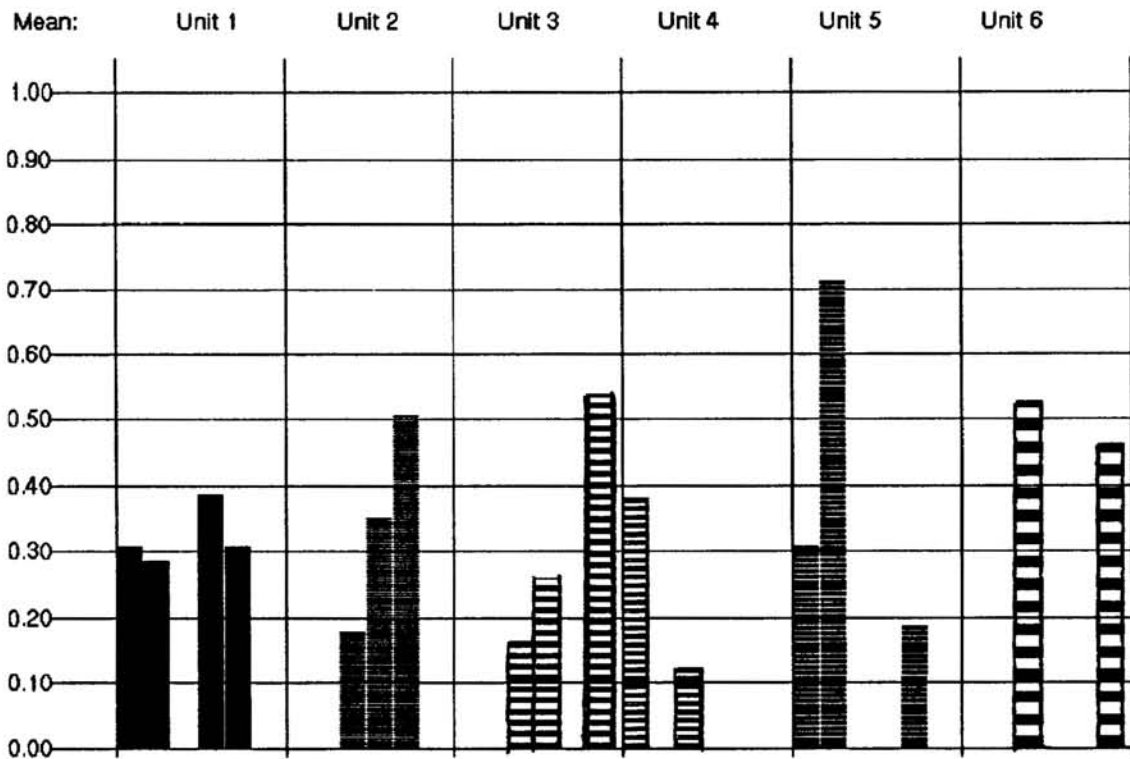

b)

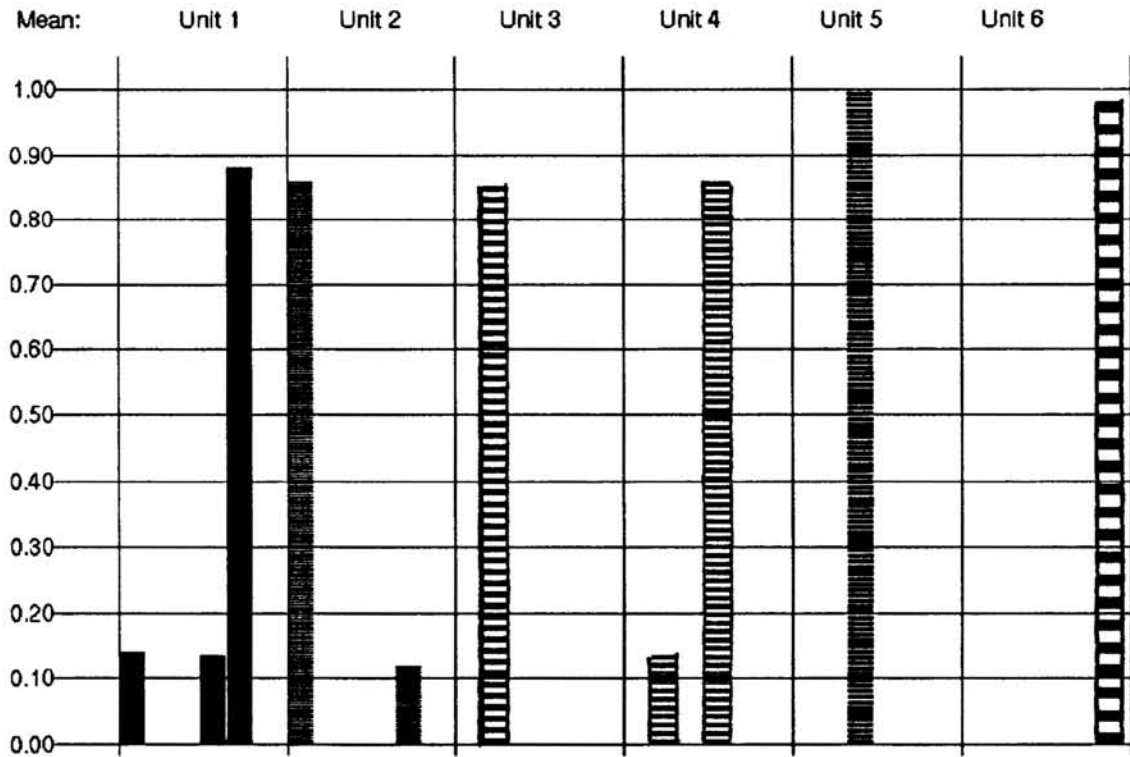

Figure 2: *The probability of each output unit responding for each of the six classes of patterns, averaged over 1200 cases. In a) the pattern trajectories contained six shifted examples of each class, while in b) there were three examples of each class.*

the patterns. Additionally, because we have a well-defined objective function for the learning, the procedure can be applied to multi-layer networks which discover features specifically tuned to the classification problem.

However, there are a few drawbacks to using this learning procedure. One is that if any lower-order temporally coherent structure exists, the network will invariably discover it. So, for example, if the pattern classes differ in their average intensity, the network can easily learn to separate them simply by detecting the average intensity of the inputs and ignoring all other information. Similarly, if the spatial location of pattern features varies slowly and predictably over time, the network tends to learn a spatial map rather than solving the higher-order problem of pattern classification. On the other hand, this suggests that a sequential approach to modelling temporally coherent structure may be possible: an initial processing stage could try to model low-order temporal structure such as local spatial correlations, a second processing stage could model the remaining structure in the output of the first over a larger spatio-temporal extent, and so on.

A second drawback is the space complexity of the algorithm: for a network with $n$ output units, each must store $n^2$ joint probability statistics and $n$ individual probabilities.[2] The storage complexity can be reduced from $n^2 + n$ to just two statistics per output unit by optimizing a more constrained objective function in which each output unit assumes a maximum entropy distribution for the other $n-1$ units. It then need only consider the average probability of its own output, and the joint probability of its output at successive time steps. In this case, the mutual information can be approximated by a sum of $n$ terms:

$$I_{C_t;C_{t+1}} \simeq \sum_i I_{c_{i,t};c_{i,t+1}}$$

$$= \sum_i H(c_{i,t}) + H(c_{i,t+1}) - H(c_{i,t}, c_{i,t+1})$$

where $H(c_{i,t}) = -\langle p_i{}^t \rangle_t \log \langle p_i{}^t \rangle_t - \frac{n-1}{n} \langle 1 - p_i{}^t \rangle_t \log \frac{n-1}{n} \langle 1 - p_i{}^t \rangle_t$ is the entropy of the $i$th output unit under the maximum entropy assumption for the other output units, and the other constrained entropies are computed similarly.

A final drawback of the learning procedure presented here, as discussed earlier, is its tendency to become trapped in local optima with very large weights. We dealt with this by introducing a constant parameter, $\lambda$, to dampen the importance of the joint entropy term. A more principled way to deal with the problem of local optima is to use stochastic rather than deterministic output units, resulting in a stochastic gradient descent learning procedure (although this would increase the simulation time considerably). Another way of obtaining more globally optimal solutions might be to consider the predictability of classifications over longer time scales rather than just at pairwise time steps, as was done in Földiák's model (1991). The network could thus maximize the mutual information between its current response and a weighted average of its responses over the last few time steps.

## 5   CONCLUSIONS

The invariance of an objects' identity over time, with respect to transformations it may undergo as it and/or the observer move, provides a powerful cue for perceptual learning. We have demonstrated that a network can learn, entirely unsupervised, to build translation-invariant object detectors based on the assumption of temporal coherence about the input. This procedure should be widely applicable to a variety of perceptual learning tasks, such as identifying phonemes in speech, segmenting objects in images of trajectories, and classifying textures in tactile input.

**Acknowledgments**

I thank Geoff Hinton for many fruitful discussions that led to the ideas presented in this paper.

## Footnotes

*Address as of July 1993: Department of Psychology, McMaster University, 1280 Main Street West, Hamilton Ontario, Canada, L8S 4K1

[1] Removing biases from the outputs helps prevent the network from getting trapped in local maxima during learning.

[2]Note, however, that the complexity (both in time and space) of the computation of these statistics is negligible relative to that of the gradient calculations, assuming there are many more weights than the squared number of output units in the network.

## References

Becker, S. and Hinton, G. E. (1992). A self-organizing neural network that discovers surfaces in random-dot stereograms. *Nature*, 355:161–163.

Birch, E. E., Gwiazda, J., and Held, R. (1982). Stereoacuity development for crossed and uncrossed disparities in human infants. *Vison research*, 22:507–513.

Bridle, J. S. (1990). Training stochastic model recognition algorithms as networks can lead to maximum mutual information estimation of parameters. In Touretzky, D. S., editor, *Neural Information Processing Systems, Vol. 2*, pages 111–217, San Mateo, CA. Morgan Kaufmann.

Földiák, P. (1991). Learning invariance from transformation sequences. *Neural Computation*, 3(2):194–200.

Fukushima, K. (1988). Neocognition: A hierarchical neural network capable of visual pattern pattern recognition. *Neural networks*, 1:119–130.

Giles, C. L. and Maxwell, T. (1987). Learning, invariance, and generalization in high-order neural networks. *Applied Optics*, 26(23):4972–4978.

Held, R., Birch, E. E., and Gwiazda, J. (1980). Stereoacuity of human infants. *Proceedings of the national academy of sciences USA*, 77(9):5572–5574.

Hinton, G. E. and Lang, K. (1985). Shape recognition and illusory conjunctions. In *IJCAI 9*, Los Angeles.

Le Cun, Y., Boser, B., Denker, J., Henderson, D., Howard, R., Hubbard, W., and Jackel, L. (1990). Handwritten digit recognition with a back-propagation network. In Touretzky, D., editor, *Advances in Neural Information Processing Systems*, pages 396–404, Denver 1989. Morgan Kaufmann, San Mateo.

Zemel, R. S., Hinton, G. E., and Mozer, M. C. (1990). TRAFFIC: object recognition using hierarchical reference frame transformations. In *Advances in Neural Information Processing Systems 2*, pages 266–273. Morgan Kaufmann Publishers.
